# Inferring spike-timing-dependent plasticity from spike train data

**Ian H. Stevenson and Konrad P. Kording**
Department of Physical Medicine and Rehabilitation
Northwestern University
`{i-stevenson, kk}@northwestern.edu`

## Abstract

Synaptic plasticity underlies learning and is thus central for development, memory, and recovery from injury. However, it is often difficult to detect changes in synaptic strength in vivo, since intracellular recordings are experimentally challenging. Here we present two methods aimed at inferring changes in the coupling between pairs of neurons from extracellularly recorded spike trains. First, using a generalized bilinear model with Poisson output we estimate time-varying coupling assuming that all changes are spike-timing-dependent. This approach allows model-based estimation of STDP modification functions from pairs of spike trains. Then, using recursive point-process adaptive filtering methods we estimate more general variation in coupling strength over time. Using simulations of neurons undergoing spike-timing dependent modification, we show that the true modification function can be recovered. Using multi-electrode data from motor cortex we then illustrate the use of this technique on in vivo data.

## 1 Introduction

One of the fundamental questions in computational neuroscience is how synapses are modified by neural activity [1, 2]. A number of experimental results, using intracellular recordings in vitro, have shown that synaptic plasticity depends on the precise pairing of pre- and post-synaptic spiking [3]. While such spike-timing-dependent plasticity (STDP) is thought to serve as a powerful regulatory mechanism [4], measuring STDP in vivo using intracellular recordings is experimentally difficult [5]. Here we instead attempt to estimate STDP in vivo by using simultaneously recorded extracellular spike trains and develop methods to estimate the time-varying strength of synapses.

In the past few years model-based methods have been developed that allow the estimation of coupling between pairs of neurons from spike train data [6, 7, 8, 9, 10, 11]. These methods have been successfully applied to data from a variety of brain areas including retina [10], hippocampus [8], as well as cortex [12]. While anatomical connections between pairs of extracellularly recorded neurons are generally not guaranteed, these phenomenological methods regularly improve encoding accuracy and provide a statistical description of the functional coupling between neurons.

Here we present two techniques that extend these statistical methods to time-varying coupling between neurons and allow the estimation of spike-timing-dependent plasticity from spike trains. First we introduce a generative model for time-varying coupling between neurons where the changes in coupling strength depend on the relative timing of pre- and post-synaptic spikes: a bilinear-nonlinear-Poisson model. We then present two approaches for inferring STDP modification functions from spike data. We test these methods on both simulated data and data recorded from the motor cortex of a sleeping macaque monkey.

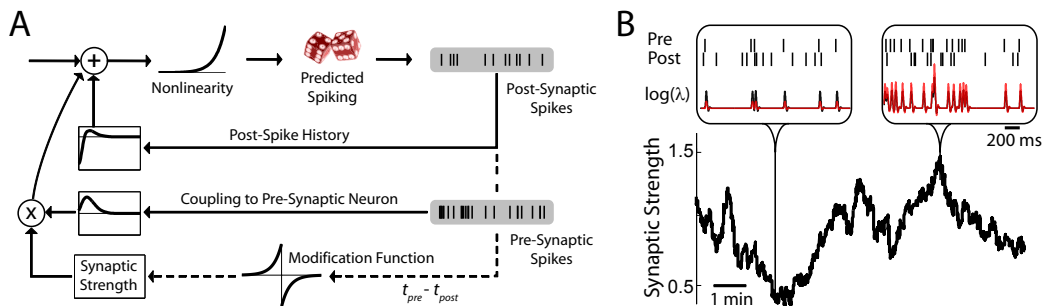

Figure 1: Generative model. A) A generative model of spikes where the coupling between neurons undergoes spike-timing dependent modification. Post-synaptic spiking is modeled as a doubly stochastic Poisson process with a conditional intensity that depends on the neuron's own history and coupling to a pre-synaptic neuron. We consider the case where the strength of the coupling changes over time, depending on the relative timing of pre- and post-synaptic spikes through a modification function. B) As the synaptic strength changes over time, the influence of the pre-synaptic neuron on the post-synaptic neuron changes. Insets illustrate two points in time where synaptic strength is low (left) and high (right), respectively. Red lines illustrate the time-varying influence of the pre-synaptic neuron, while the black lines denote the static influence.

## 2   Methods

Many studies have examined nonstationarity in neural systems, including for decoding [13], unitary event detection [14], and assessing statistical dependencies between neurons [15]. Here we focus specifically on non-stationarity in coupling between neurons due to spike-timing dependent modification of synapses. Our aim is to provide a framework for inferring spike-timing dependent modification functions from spike train data alone. We first present a generative model for spike trains where neurons are undergoing STDP. We then present two methods for estimating spike-timing dependent modification functions from spike train data: a direct method based on a time-varying generalized linear model (GLM) and an indirect method based on point-process adaptive filtering.

### 2.1   A generative model for coupling with spike-timing dependent modification

While STDP has traditionally been modeled using integrate-and-fire neurons [4, 16], here we model neurons undergoing STDP using a simple rate model of coupling between neurons, a linear-nonlinear-Poisson (LNP) model. In our LNP model, the conditional intensity (instantaneous firing rate) of a neuron is given by a linear combination of covariates passed through a nonlinearity. Here, we assume that this nonlinearity is exponential, and the LNP reduces to generalized linear model (GLM) with a canonical log link function.

The covariates driving variations in the neuron's firing rate can depend on the past spiking history of the neuron, the past spiking history of other neurons (coupling), as well as any external covariates such as visual stimuli [10] or hand movement [12]. To model coupling from a pre-synaptic neuron to a post-synaptic neuron, here we assume that the post-synaptic neuron's firing is generated by

$$\lambda(t \mid \boldsymbol{H}_t, \boldsymbol{\alpha}, \boldsymbol{\beta}) = \exp\left( \alpha_0 + \sum_i f_i \left( n_{post}(t - \tau : t) \right) \alpha_i + \sum_j f_j(n_{pre}(t - \tau : t)) \beta_j \right)$$
$$n_{post}(t) \sim Poisson(\lambda(t \mid \boldsymbol{H}_t, \boldsymbol{\alpha}, \boldsymbol{\beta})\Delta t) \tag{1}$$

where $\lambda(t \mid \boldsymbol{H}_t, \boldsymbol{\alpha}, \boldsymbol{\beta})$ is the conditional intensity of the post-synaptic neuron at time $t$, given a short history of past spikes from the two neurons $\boldsymbol{H_t}$ and the model parameters. $\alpha_0$ defines a baseline firing rate, which is modulated by both the neuron's own spike history from $t-\tau$ to $t$, $n_{post}(t-\tau : t)$, and the history of the pre-synaptic neuron $n_{pre}(t - \tau : t)$ (together abbreviated as $\boldsymbol{H_t}$). Here we have assumed that the post-spike history and coupling effects are mapped into a smooth basis by a set of functions $f_i$ and then weighted by a set of post-spike coefficients $\boldsymbol{\alpha}$ and a set of coupling

coefficients $\boldsymbol{\beta}$. Finally, we assume that spikes $n_{post}(t)$ are generated by a Poisson random variable with rate $\lambda(t \mid \boldsymbol{H}_t, \boldsymbol{\alpha}, \boldsymbol{\beta})\Delta t$.

This model has been used extensively over the past few years to model coupling between neurons [10, 12]. Details and extensions of this basic form have been previously published [6]. It is important to note, however, that the parameters $\boldsymbol{\alpha}$ and $\boldsymbol{\beta}$ can be easily estimated by maximizing the log-likelihood. Since the likelihood is log-concave [9], there is a single, global solution which can be found quickly by a number of methods, such as iterative reweighted least squares (IRLS, used here).

Here we consider the case where the coupling strength can vary over time, and particularly as a function of precise timing between pre- and post-synaptic spikes. To incorporate these spike-timing dependent changes in coupling into the generative model we introduce a time-varying coupling strength or "synaptic weight" $w(t)$

$$\lambda(t \mid \boldsymbol{X}, \boldsymbol{\alpha}, \boldsymbol{\beta}) = \exp\left(\alpha_0 + \boldsymbol{X}_s(t)\boldsymbol{\alpha} + w(t)\boldsymbol{X}_c(t)\boldsymbol{\beta}\right)$$
$$n_{post}(t) \sim Poisson(\lambda(t \mid \boldsymbol{X}, \boldsymbol{\alpha}, \boldsymbol{\beta})\Delta t) \tag{2}$$

where $w(t)$ changes based on the relative timing of pre- and post-synaptic spikes. Here, for simplicity, we have re-written the stable post-spike history and coupling terms in matrix form. The vector $\boldsymbol{X}_s(t)$ summarizes the post-spike history covariates at time $t$ while $\boldsymbol{X}_c(t)$ summarizes the covariates related to the history of the pre-synaptic neuron. In this model, the synaptic weight $w(t)$ simply acts to scale the stable coupling defined by $\boldsymbol{\beta}$, and we update $w(t)$ such that every pre-post spike pair alters the synaptic weight independently following the second spike.

Under this model, the firing rate of the post-synaptic neuron is influenced by it's own past spiking, as well as the activity of a pre-synaptic neuron. A synaptic weight determining the strength of coupling between the two neurons changes over time depending on the relative spike-timing (Fig 1A).

In the simulations that follow we consider three types of modification functions: 1) a traditional double-exponential function that accurately models STDP found in cortical and hippocampal slices, 2) a mexican-hat type function that qualitatively matches STDP found in GABA-ergic neurons in hippocampal cultures, and 3) a smoothed double-exponential function that has recently been demonstrated to stabilize weight distributions [17].

The double-exponential modification function is consistent with original STDP observations [2, 3] and has been used extensively in simulated populations of integrate-and-fire neurons [4, 16]. In this case each pair of pre- and post-synaptic spikes modifies the synapse by

$$\Delta w(t_{pre} - t_{post}) = \begin{cases} A_+ \exp\left(\frac{t_{pre}-t_{post}}{\tau_+}\right) & \text{if } t_{pre} < t_{post} \\ A_- \exp\left(-\frac{t_{pre}-t_{post}}{\tau_-}\right) & \text{if } t_{pre} \geq t_{post} \end{cases} \tag{3}$$

where $t_{pre}$ and $t_{post}$ denote the relative spike times, and the parameters $A_+$, $A_-$, $\tau_+$, and $\tau_-$ determine the magnitude and drop-off of each side of the double-exponential. This creates a sharp boundary where the synapse is strengthened whenever pre-synaptic spikes appear to cause post-synaptic spikes and weakened when post-synaptic spikes do not immediately proceed pre-synaptic spikes.

Similarly, mexican-hat type functions qualitatively match observations of STDP in GABA-ergic neurons in hippocampal cultures [18] where

$$\Delta w(t_{pre} - t_{post}) = A_+ \exp\left(\frac{-(t_{pre} - t_{post})^2}{2\tau_+^2}\right) + A_- \exp\left(\frac{-(t_{pre} - t_{post})^2}{2\tau_-^2}\right) \tag{4}$$

For $\tau_- > \tau_+$ this corresponds to a more general Hebbian rule, where synapses are strengthened whenever pre- and post-synaptic spikes occur in close proximity. When spikes do not occur in close proximity the synapse is weakened. In this case, the parameters $A_+$, $A_-$, $\tau_+$, and $\tau_-$ determine the magnitude and standard deviation of the positive and negative components of the modification function.

Finally, we consider a smoothed double-exponential modification function that has recently been shown to stabilize weight distributions. The sharp causal boundary in the classical double-exponential tends to drive synaptic weights either towards a maximum or to zero. By adding noise

to $t_{pre} - t_{post}$, this causal boundary can be smoothed and weight distributions become stable [17]. Here we add Gaussian noise to (3) such that $(t_{pre} - t_{post})' = (t_{pre} - t_{post}) + \epsilon, \epsilon \sim N(0, \sigma^2)$.

It is important to note that, unlike more common integrate-and-fire models of STDP, these modification function do not describe a change in the magnitude of post synaptic potentials (PSPs). Rather, $\Delta w$ defines a change in the statistical influence of the pre-synaptic neuron on the post-synaptic neuron. When $w(t) \boldsymbol{X}_c(t) \boldsymbol{\beta}$ is large, the post-synaptic neuron is more likely to fire following a pre-synaptic spike. However, in this bilinear form, $w(t)$ is only uniquely defined up to a multiplicative constant.

This generative model includes two distinct components: a GLM that defines the stationary firing properties of the post-synaptic neuron and a modification function that defines how the coupling between the pre- and post-synaptic neuron changes over time as a function of relative spike timing. In simulating isolated pairs of neurons, each of the modification functions described above induces large variations in the synaptic weight. For the sake of stable simulation we add an additional long-timescale forgetting factor that pushes the synaptic weights back to 1. Namely,

$$w(t + \Delta t) = \begin{cases} w(t) - \frac{\Delta t}{\tau_f}(w(t) - 1) + \Delta w(t_{pre} - t_{post}) & \text{if } n_{pre} \text{ or } n_{post} = 1 \\ w(t) - \frac{\Delta t}{\tau_f}(w(t) - 1) & \text{otherwise} \end{cases} \tag{5}$$

where, here, we use $\tau_f = 60s$. The next sections describe two methods for estimating time-varying synaptic strength as well as STDP modification functions from spike train data.

## 2.2 Point-process adaptive filtering of coupling strength

Several recent studies have examined the possibility that the tuning properties of neurons may drift over time. In this context, techniques for estimating arbitrary changes in the parameters of LNP models have been especially useful. Point-process adaptive filtering is one such method which allows accurate estimation of arbitrary time-varying parameters within LNP models and GLMs [19, 20]. The goal of this filtering approach is to update the model parameters at each time step, following spike observations, based on the instantaneous likelihood. Here we use this approach to track variations in coupling strength between two neurons over time.

Details and a complete derivation of this model have been previously presented [20]. Briefly, the basic recursive point-process adaptive filter follows a standard state-space modeling approach and assumes that the model parameters in a GLM, such as (1), vary according to a random walk

$$\boldsymbol{\beta}_{t+1} = \boldsymbol{F_t} \boldsymbol{\beta}_t + \boldsymbol{\eta}_t \tag{6}$$

where $\boldsymbol{F_t}$ denotes the transition matrix from one timestep to the next and $\boldsymbol{\eta}_t \sim N(0, \boldsymbol{Q}_t)$ denotes Gaussian noise with covariance $\boldsymbol{Q}_t$. Given this state-space assumption, we can update the model parameters $\boldsymbol{\beta}$ given incoming spike observations. The prediction density at each timestep is given by

$$\boldsymbol{\beta}_{t|t-1} = \boldsymbol{F_t} \boldsymbol{\beta}_{t-1|t-1}$$
$$\boldsymbol{W}_{t|t-1} = \boldsymbol{F_t} \boldsymbol{W}_{t-1|t-1} \boldsymbol{F}_t^T + \boldsymbol{Q}_t \tag{7}$$

where $\boldsymbol{\beta}_{t-1|t-1}$ and $\boldsymbol{W}_{t-1|t-1}$ denote the estimated mean and covariance from the previous timestep. Given a new spike count observation $n_t$, we then integrate this prior information with the likelihood to obtain the posterior. Here, for simplicity, we use a quadratic expansion of the log-posterior (a Laplace approximation). When $\log \lambda$ is linear in the parameters, the conditional intensity and posterior are given by

$$\lambda_t = \exp\left(\boldsymbol{X}_t \boldsymbol{\beta}_{t|t-1} + c_t\right)$$
$$\boldsymbol{W}_{t|t}^{-1} = \boldsymbol{W}_{t|t-1}^{-1} + \boldsymbol{X}_t^T [\lambda_t \Delta t] \boldsymbol{X}_t$$
$$\boldsymbol{\beta}_{t|t} = \boldsymbol{\beta}_{t|t-1} + \boldsymbol{W}_{t|t} \left[\boldsymbol{X}_t^T (n_t - \lambda_t \Delta t)\right] \tag{8}$$

where $\boldsymbol{X}_t$ denotes the covariates corresponding to the state-space variable, and $c_t$ describes variation in $\log \lambda$ that is assumed to be stable over time. Here, the state-space variable is coupling strength,

and stable components of the model, such as post-spike history effects, are summarized with $c_t$. The initial values of $\boldsymbol{\beta}$ and $\boldsymbol{W}$ can be estimated using a short training period before filtering. The only free parameters are those describing the state-space: $\boldsymbol{F}$ and $\boldsymbol{Q}$. In the analysis that follows we will reduce the problem to a single dimension, where the shape of coupling is fixed during training, and we apply the point-process adaptive filter to a single coefficient for the covariate $X'(t) = X_c(t)\boldsymbol{\beta}$.

Together, (7) and (8) allow us to track changes in the model parameters over time. Given an estimate of the time-varying synaptic weight $\hat{w}(t)$, we can then estimate the modification function $\Delta\hat{w}(t_{pre} - t_{post})$ by correlating the estimated changes in $\hat{w}(t)$ with the relative spike timings that we observe.

### 2.3 Inferring STDP with a nonparametric, generalized bilinear model

Point-process adaptive filtering allows us to track noisy changes in coupling strength over time. However, it does not explicitly model the fact that these changes may be spike-timing dependent. In this section we introduce a method to directly infer modification functions from spike train data. Specifically, we model the modification function non-parametrically by generating covariates $\boldsymbol{W}$ that depend on the relative spike timing. This non-parametric approximation to the modification gives a generalized bilinear model (GBLM).

$$\lambda(t \mid \boldsymbol{X}, \boldsymbol{W}, \boldsymbol{\alpha}, \boldsymbol{\beta}, \boldsymbol{\beta}_w) = \exp\left(\alpha_0 + \boldsymbol{X}_s(t)\boldsymbol{\alpha} + \boldsymbol{\beta}_w^T \boldsymbol{W}^T(t)\boldsymbol{X}_c(t)\boldsymbol{\beta}\right)$$

$$n_{post}(t) \sim Poisson(\lambda(t \mid \boldsymbol{X}, \boldsymbol{W}, \boldsymbol{\alpha}, \boldsymbol{\beta}, \boldsymbol{\beta}_w)\Delta t) \tag{9}$$

where $\boldsymbol{\beta}_w$ describes the modification function and $\boldsymbol{W}(t)\boldsymbol{\beta}_w$ approximates $w(t)$. Each of the $K$ STDP covariates, $\boldsymbol{W}_k$, describes the cumulative effect of spike pairs $t_{pre} - t_{post}$ within a specific range $[T_k^-, T_k^+]$,

$$W_k(t + \Delta t) = W_k(t) - \frac{\Delta t}{\tau_f}(W_k(t) - 1) + \mathbf{1}(t_{pre} - t_{post} \in [T_k^-, T_k^+]) \tag{10}$$

such that, together, $\boldsymbol{W}(t)\boldsymbol{\beta}_w$ captures the time-varying coupling due to pre-post spike pairs within a given window (i.e. -100 to 100ms). To model any decay in STDP over time, we, again, allow these covariates to decay exponentially with $\tau_f$.

In this form, maximum likelihood estimation along each axis is a log-concave optimization problem [21]. The parameters describing the modification function $\boldsymbol{\beta}_w$ and the parameters describing the stable parts of the model $\boldsymbol{\alpha}$ and $\boldsymbol{\beta}$ can be estimated by holding one set of parameters fixed while updating the other and alternating between the two optimizations. In practice, convergence is relatively fast, with the deviance changing by $< 0.1\%$ within 3 iterations (Fig 3A), and, empirically, using random restarts, we find that the solutions tend to be stable. In addition to estimates of the post-spike history and coupling filters, the GBLM thus provides a non-parametric approximation to the modification function and explicitly accounts for spike-timing dependent modification of the coupling strength.

## 3 Results

To examine the accuracy and convergence properties of the two inference methods presented above, we sampled spike trains from the generative model with various parameters. We simulated a pre-synaptic neuron as a homogeneous Poisson process with a firing rate of $5Hz$, and the post-synaptic neuron as a conditionally Poisson process with a baseline firing rate of $5Hz$. Through the GBLM, the post-synaptic neuron's firing rate is affected by its own post-spike history as well as the activity of the pre-synaptic neuron (modeled using 5 raised cosine basis functions [10]). However, as STDP occurs the strength of coupling between the neurons changes according to one of three modification functions: a double-exponential, a mexican-hat, or a smoothed double-exponential (Fig 2).

We find that both point-process adaptive filtering and the generalized bilinear model are able to accurately reconstruct the time-varying synaptic weight for each type of modification function (Fig 2, left). However, adaptive filtering generally provides a much less accurate estimate of the underlying modification function than the GBLM (Fig 2, center). Since the adaptive filter only updates the

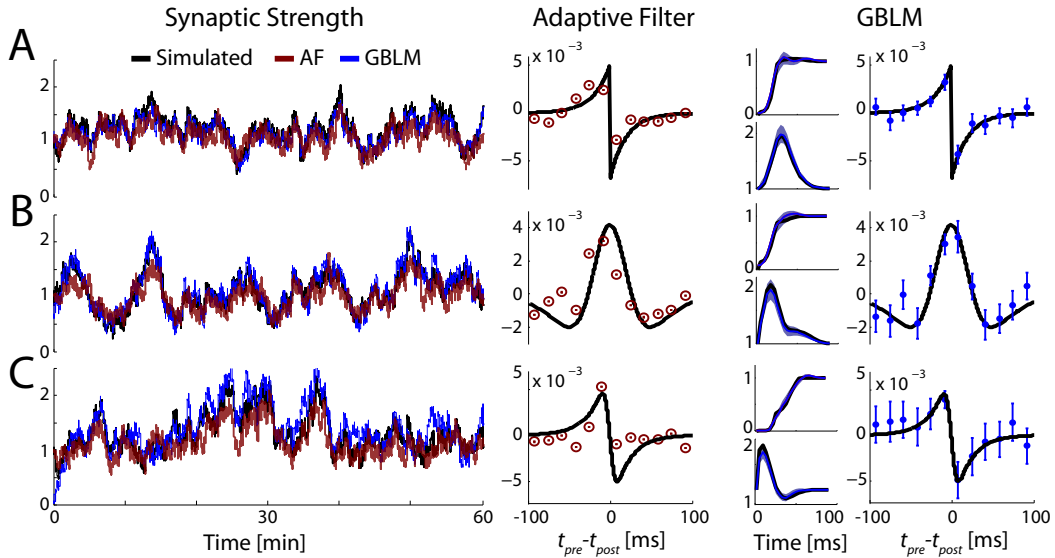

Figure 2: Recovering simulated STDP. Spikes were simulated from two neurons whose coupling varied over time, depending on the relative timing of pre- and post-synaptic spikes. Using two distinct methods (point-process adaptive filtering and the GBLM) we estimated the time-varying coupling strength and modification function from simulated spike train data. Results are shown for three different modification functions A) double-exponential, B) Mexican-hat, and C) smoothed double-exponential. Black lines denote true values, red lines denote estimates from adaptive filtering, and blue lines denote estimates from the GBLM. The post-spike history and coupling terms are shown at left for the GBLM as multiplicative gains $exp(\beta)$. Error bars denote standard errors for the post-spike and coupling filters and 95% confidence intervals for the modification function estimates.

synaptic weight following the observations $n_t$, this is not entirely unsurprising. Changes in coupling strength are only detected by the filter after they have occurred and become evident in the spiking of the post-synaptic neuron. In contrast to the GBLM, there is a substantial delay between changes in the true synaptic weight and those estimated by the adaptive filter. In this case, we find that the accuracy of the adaptive filter follows changes in the synaptic weight approximately exponentially with $\tau \sim 25ms$ (Fig 3B).

An important question for the practical application of these methods is how much data is necessary to detect and accurately estimate modification functions for various effect sizes. Since the size of spike-timing dependent changes may be small in vivo, it is essential that we know under which conditions modification functions can be recovered. Here we simulated the standard double-exponential STDP model with several different effect-sizes, modifying $A_+$ and $A_-$ and examining the estimation error in both $\hat{w}(t)$ and $\Delta\hat{w}(t_{pre} - t_{post})$ (Fig 3). The three different effect-sizes simulated here used coupling kernels similar to Fig 2A and began with $w(t) = 1$. After spike simulation the standard deviation in $w(t)$ was 0.060±0.002 for the small effect size, 0.13±0.01 for the medium effect size, and 0.27±0.01 for the large effect size. For all effect sizes, we found that with small amounts of data ($< 1000$ s), the GBLM tends to over-fit the data. In these situations Adaptive Filtering reconstructs both the synaptic weight (Fig 3E) and modification function (Fig 3F) more accurately than the GBLM (Fig 3C,E). However, once enough data is available maximum likelihood estimation of the GBLM out-performs both the stable coupling model and adaptive filtering. The extent of over-fitting can be assessed by the cross-validated log likelihood ratio relative to the homogeneous Poisson process (Fig 3G, shown in $\log_2$ for 2-fold cross-validation). Here, the stable coupling model has an average cross-validated log likelihood ratio relative to a homogeneous Poisson process of 0.185± 0.004 bits/spike across all effect sizes. Even in this controlled simulation the contribution of time-varying coupling is relatively small. Both the GBLM and Adaptive Filtering only increase the log likelihood relative to a homogeneous Poisson process by 3-4% for the parameters used here at the largest recording length.

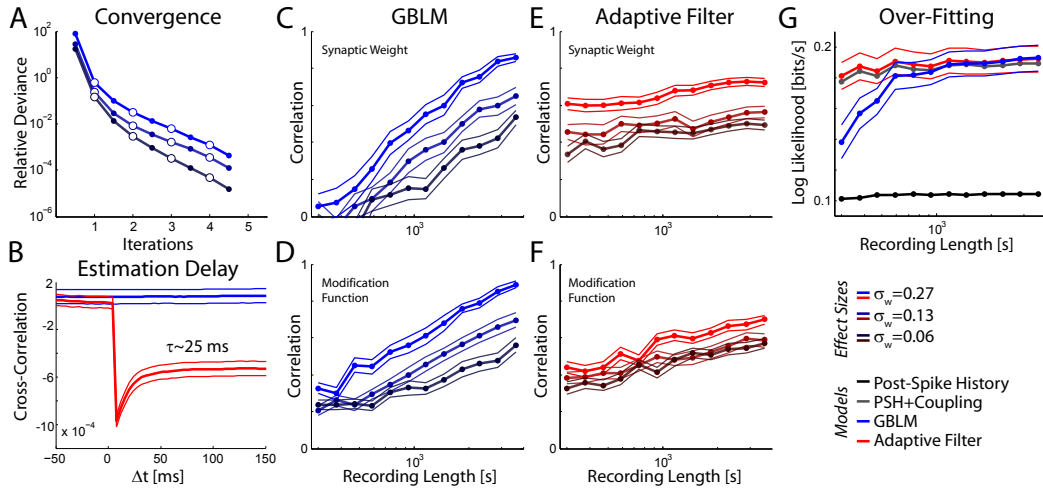

Figure 3: Estimation errors for simulated STDP. A) Convergence of the joint optimization problem for three different effect sizes. Filled circles denote updates of the stable coupling terms. Open circles denote updates of the modification function terms. Note that after 3 iterations the deviance is changing by $< 0.1\%$ and the model has (essentially) converged. B) Cross-correlation between changes in the true synaptic weight and estimated weight for the GBLM and Adaptive Filter. Note that Adaptive Filtering fails to predict weight changes as they occur. Error bars denote SEM across N=10 simulations at the largest effect size. C,D) Correlation between the simulated and estimated synaptic weight (C) and modification function (D) for the GBLM as a function of the recording length. E,F) Correlation between the simulated and estimated synaptic weight and modification function for Adaptive Filtering. Error bars denote SEM across N=40 simulations for each effect size. G) Cross-validated (2-fold) log likelihood relative to a homogeneous Poisson process for the GBLM and Adaptive Filtering models. The GBLM (blue) over-fits for small amounts of data, but eventually out-performs both the stable coupling model (gray) and Adaptive Filtering (red). Error bands denote SEM across N=120 simulations, all effect sizes.

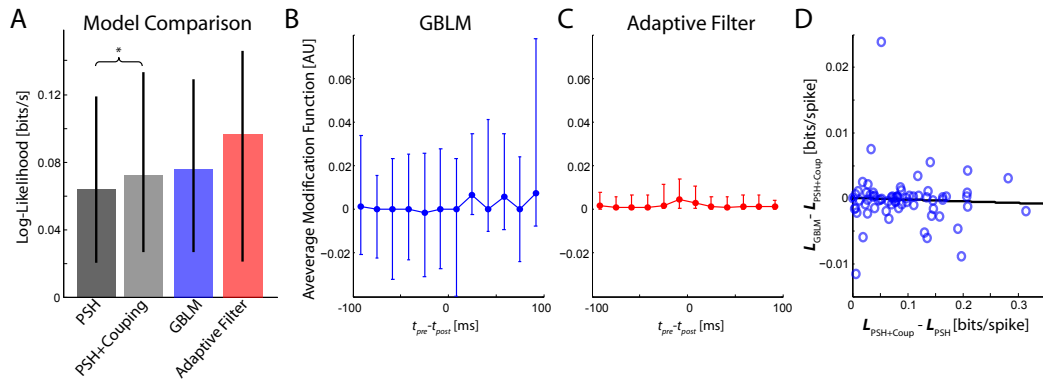

Figure 4: Results for data from monkey motor cortex. A) Log likelihood relative to a homogeneous Poisson process for each of four models: a stable GLM with only post-spike history (PSH), a stable GLM with PSH and coupling, the GBLM, and the Adaptive Filter. Bars and error bars denote median and inter-quartile range. * denotes significance under a paired t-test, p<0.05. B) The average modification function estimated under the GBLM for N=75 pairs of neurons. C) The modification function estimated from adaptive filtering for the same data. In both cases there does not appear to be a strong, stereotypically shaped modification function. D) The degree to which adding nonstationary coupling improves model accuracy does not appear to be related to coupling strength as measured by how much the PSH+Coupling model improves model accuracy over the PSH model.

Finally, to test these methods on actual neural recordings, we examined multi-electrode recordings from the motor cortex of a sleeping macaque monkey. The experimental details of this task have been previously published [22]. Approximately 180 minutes of data from 83 neurons were collected (after spike sorting) during REM and NREM sleep.

In the simulations above we assumed that the forgetting factor $\tau_f$ was known. For the GBLM $\tau_f$ determines the timescale of the spike-timing dependent covariates $\boldsymbol{X}_w$, while for adaptive filtering $\tau_f$ defines the transition matrix $\boldsymbol{F}$. In the analysis that follows we make the simplifying assumption that the forgetting factor is fixed at $\tau_f = 60s$. Additionally, during adaptive filtering we fit the variance of the process noise $\boldsymbol{Q}$ by maximizing the cross-validated log-likelihood.

Analyzing the most strongly correlated 75 pairs of neurons during the 180 minute recording (2-fold cross-validation) we find that the GBLM and Adaptive Filtering both increase model accuracy (Fig 4A). However, the resulting modification functions do not show any of the structure previously seen in intracellular experiments. In both individual pairs and the average across pairs (Fig 4B,C) the modification functions are noisy and generally not significantly different from zero. Additionally, we find that the increase in model accuracy provided by adding non-stationary coupling to the traditional, stable coupling GLM does not appear to be correlated with the strength of coupling itself. These results suggest that STDP may be difficult to detect in vivo, requiring even longer recordings or, possibly, different electrode configurations. Particularly, with the electrode array used here (Utah array, 400 $\mu m$ electrode spacing), neurons are unlikely to be mono-synaptically connected.

## 4   Discussion

Here we have presented two methods for estimating spike-timing dependent modification functions from multiple spike train data: an indirect method based on point-process adaptive filtering and a direct method using a generalized bilinear model. We have shown that each of these methods is able to accurately reconstruct both ongoing fluctuations in synaptic weight and modification functions in simulation. However, there are several reasons that detecting similar STDP in vivo may be difficult. In vivo, pairs of neurons do not act in isolation. Rather, each neuron receives input from thousands of other neurons, inputs which may confound estimation of the coupling between a given pair. It would be relatively straightforward to include multiple pre-synaptic neurons in the model using either stable coupling [6, 10] or time-varying, spike-timing dependent coupling. Additionally, unobserved common input or external covariates, such as hand position, could also be included in the model. These extra covariates should further improve spike prediction accuracy, and could, potentially, result in better estimation of STDP modification functions.

Despite these caveats the statistical description of time-varying coupling presented here shows promise. Although the neurons in vivo are not guaranteed to be anatomically connected and estimated coupling must be always be interpreted cautiously [11], including synaptic modification terms does improve model accuracy on in vivo data. Several experimental studies have even suggested that understanding plasticity may not require well-isolated pairs of neurons. The effects of STDP may be visible through poly-synaptic potentiation [23, 24, 25]. In analyzing real data our ability to detect STDP may vary widely across experimental preparations. For instance, recordings from hippocampal slice or dissociated neuronal cultures may reveal substantially more plasticity than in vivo cortical recordings and are less likely to be confounded by unobserved common-input.

There are a number of extensions to the basic Adaptive Filtering and GBLM frameworks that may yield more accurate estimation and more biophysically realistic models of STDP. The over-fitting observed in the GBLM could be reduced by regularizing the modification function, and Adaptive Smoothing (using both forward and backward updates) will likely out-perform Adaptive Filtering as used here. By changing the functional form of the covariates included in the GBLM we may be able to distinguish between standard models of STDP where spike pairs are treated independently and other models such as those with self-normalization [16] or where spike triplets are considered [26]. Ultimately, the framework presented here extends recent GLM-based approaches to modeling coupling between neurons to allow for time-varying coupling between neurons and, particularly, changes in coupling related to spike-timing dependent plasticity. Although it may be difficult to resolve the small effects of STDP in vivo, both improvements in recording techniques and statistical methods promise to make the observation of these ongoing changes possible.

# References

[1] LF Abbott and SB Nelson. Synaptic plasticity: taming the beast. *Nature Neuroscience*, 3:1178–1183, 2000.

[2] G Bi and M Poo. Synaptic modification by correlated activity: Hebb's postulate revisited. *Annual Review of Neuroscience*, 24(1):139–166, 2001.

[3] H Markram, J Lubke, M Frotscher, and B Sakmann. Regulation of synaptic efficacy by coincidence of postsynaptic aps and epsps. *Science*, 275(5297):213–215, 1997.

[4] S Song, KD Miller, and LF Abbott. Competitive hebbian learning through spike-timing-dependent synaptic plasticity. *Nature Neuroscience*, 3(9):919–926, 2000.

[5] V Jacob, DJ Brasier, I Erchova, D Feldman, and D.E Shulz. Spike timing-dependent synaptic depression in the in vivo barrel cortex of the rat. *The Journal of Neuroscience*, 27(6):1271, 2007.

[6] Z Chen, D Putrino, S Ghosh, R Barbieri, and E Brown. Statistical inference for assessing functional connectivity of neuronal ensembles with sparse spiking data. *Neural Systems and Rehabilitation Engineering, IEEE Transactions on*, (99):1–1, 2010.

[7] S Gerwinn, JH Macke, M Seeger, and M Bethge. Bayesian inference for spiking neuron models with a sparsity prior. *Advances in Neural Information Processing Systems*, 20, 2007.

[8] M Okatan, MA Wilson, and EN Brown. Analyzing functional connectivity using a network likelihood model of ensemble neural spiking activity. *Neural Computation*, 17(9):1927–1961, 2005.

[9] L Paninski. Maximum likelihood estimation of cascade point-process neural encoding models. *Network: Computation in Neural Systems*, 15:243–262, 2004.

[10] JW Pillow, J Shlens, L Paninski, A Sher, AM Litke, EJ Chichilnisky, and EP Simoncelli. Spatio-temporal correlations and visual signalling in a complete neuronal population. *Nature*, 454(7207):995–999, 2008.

[11] IH Stevenson, JM Rebesco, LE Miller, and KP Kording. Inferring functional connections between neurons. *Current Opinion in Neurobiology*, 18(6):582–588, 2008.

[12] W. Truccolo, U. T. Eden, M. R. Fellows, J. P. Donoghue, and E. N. Brown. A point process framework for relating neural spiking activity to spiking history, neural ensemble, and extrinsic covariate effects. *Journal of Neurophysiology*, 93(2):1074–1089, 2005.

[13] W Wu and NG Hatsopoulos. Real-time decoding of nonstationary neural activity in motor cortex. *Neural Systems and Rehabilitation Engineering, IEEE Transactions on*, 16(3):213–222, 2008.

[14] S Grun, M Diesmann, and A Aertsen. Unitary events in multiple single-neuron spiking activity: Ii. nonstationary data. *Neural Computation*, 14(1):81–119, 2002.

[15] V. Ventura, C. Cai, and R. E. Kass. Statistical assessment of time-varying dependency between two neurons. *Journal of Neurophysiology*, 94(4):2940–2947, 2005.

[16] MCW Van Rossum, GQ Bi, and GG Turrigiano. Stable hebbian learning from spike timing-dependent plasticity. *Journal of Neuroscience*, 20(23):8812, 2000.

[17] B Babadi and LF Abbott. Intrinsic stability of temporally shifted spike-timing dependent plasticity. *PLoS Comput Biol*, 6(11):e1000961, 2010.

[18] M.A. Woodin, K. Ganguly, and M. Poo. Coincident pre-and postsynaptic activity modifies gabaergic synapses by postsynaptic changes in cl-transporter activity. *Neuron*, 39(5):807–820, 2003.

[19] EN Brown, DP Nguyen, LM Frank, MA Wilson, V Solo, and A Sydney. An analysis of neural receptive field dynamics by point process adaptive filtering. *Proc Natl Acad Sci USA*, 98:12261–12266, 2001.

[20] UT Eden, LM Frank, R Barbieri, V Solo, and EN Brown. Dynamic analysis of neural encoding by point process adaptive filtering. *Neural Computation*, 16(5):971–998, 2004.

[21] MB Ahrens, L Paninski, and M Sahani. Inferring input nonlinearities in neural encoding models. *Network: Computation in Neural Systems*, 19(1):35–67, 2008.

[22] N Hatsopoulos, J Joshi, and JG O'Leary. Decoding continuous and discrete motor behaviors using motor and premotor cortical ensembles. *Journal of Neurophysiology*, 92(2):1165–1174, 2004.

[23] G Bi and M Poo. Distributed synaptic modification in neural networks induced by patterned stimulation. *Nature*, 401(6755):792–795, 1999.

[24] A Jackson, J Mavoori, and EE Fetz. Long-term motor cortex plasticity induced by an electronic neural implant. *Nature*, 444(7115):56–60, 2006.

[25] JM Rebesco, IH Stevenson, K Kording, SA Solla, and LE Miller. Rewiring neural interactions by microstimulation. *Frontiers in Systems Neuroscience*, 4:12, 2010.

[26] RC Froemke and Y Dan. Spike-timing-dependent synaptic modification induced by natural spike trains. *Nature*, 416(6879):433–438, 2002.

